# Interpreting prediction markets: a stochastic approach

**Rafael M. Frongillo**
Computer Science Divison
University of California, Berkeley
raf@cs.berkeley.edu

**Nicolás Della Penna**
Research School of Computer Science
The Australian National University
me@nikete.com

**Mark D. Reid**
Research School of Computer Science
The Australian National University & NICTA
mark.reid@anu.edu.au

## Abstract

We strengthen recent connections between prediction markets and learning by showing that a natural class of market makers can be understood as performing stochastic mirror descent when trader demands are sequentially drawn from a fixed distribution. This provides new insights into how market prices (and price paths) may be interpreted as a summary of the market's belief distribution by relating them to the optimization problem being solved. In particular, we show that under certain conditions the stationary point of the stochastic process of prices generated by the market is equal to the market's Walrasian equilibrium of classic market analysis. Together, these results suggest how traditional market making mechanisms might be replaced with general purpose learning algorithms while still retaining guarantees about their behaviour.

## 1  Introduction and literature review

This paper is part of an ongoing line of research, spanning several authors, into formal connections between markets and machine learning. In [5] an equivalence is shown between the theoretically popular prediction market makers based on sequences of proper scoring rules and follow the regularised leader, a form of no-regret online learning. By modelling the traders that demand the assets the market maker is offering we are able to extend the equivalence to stochastic mirror decent. The dynamics of wealth transfer is studied in [3], for a sequence of markets between agents that behave as Kelly bettors (i.e. have log utilities), and an equivalence to stochastic gradient decent is analysed. More broadly, [9, 2] have analysed how a wide range of machine learning models can be implemented in terms of market equilibria.

The literature on the interpretation of prediction market prices [7, 11] has had the goal of relating the equilibrium prices to the distribution of the beliefs of traders. More recent work [8] has looked at a stochastic model, and studied the behavior of simple agents sequentially interacting with the market. We continue this latter path of research, motivated by the observation that the equilibrium price may be a poor predictor of the behavior in a volitile prediction market. As such, we seek a more detailed understanding of the market than the equilibrium point – we would like to know what the "stationary distribution" of the price is, as time goes to infinity.

As is standard in the literature, we assume a fixed (product) distribution over traders' beliefs and wealth. Our model features an automated market maker, following the framework of [1] is becoming a standard framework in the field.

We obtain two results. First, we prove that under certain conditions the stationary point of our stochastic process defined by the market maker and a belief distribution of traders converges to the Walrasian equilibrium of the market as the market liquidity increases. This result, stated in Theorem 1, is general in the sense that only technical convergence conditions are placed on the demand functions of the traders – as such, we believe it is a generalisation of the stochastic result of [8] to cases where agents are are not limited to linear demands, and leave this precise connection to future work.

Second, we show in Corollary 1 that when traders are Kelly bettors, the resulting stochastic market process is equivalent to stochastic mirror descent; see e.g. [6]. This result adds to the growing literature which relates prediction markets, and automated market makers in general, to online learning; see e.g. [1], [5], [3] .

This connection to mirror descent seems to suggest that the prices in a prediction market at any given time may be meaningless, as the final point in stochastic mirror descent often has poor convergence guarantees. However, standard results suggest that a prudent way to form a "consensus estimate" from a prediction market is to *average* the prices. The average price, assuming our market model is reasonable, is provably close to the stationary price. In Section 5 we give a natural example that exhibits this behavior. Beyond this, however, Theorem 2 gives us insight into the relationship between the market liquidity and the convergence of prices; in particular it suggests that we should increase liquidity at a rate of $\sqrt{t}$ if we wish the price to settle down at the right rate.

## 2   Model

Our market model will follow the automated market maker framework of [1]. We will equip our market maker with a strictly convex function $C : \mathbb{R}^n \to \mathbb{R}$ which is twice continuously differentiable. For brevity we will write $\varphi := \nabla C$. The outcome space is $\Omega$, and the contracts are determined by a payoff function $\phi : \Omega \to \mathbb{R}^n$ such that $\Pi := \varphi(\mathbb{R}^n) = \text{ConvHull}(\phi(\Omega))$. That is, the derivative space $\Pi$ of $C$ (the "instantaneous prices") must be the convex hull of the payoffs.

A trader purchasing shares at the current prices $\pi \in \mathbb{R}^n$ pays $C(\varphi^{-1}(\pi) + r) - C(\varphi^{-1}(\pi))$ for the bundle of contracts $r \in \mathbb{R}^n$. Note that our dependence solely on $\pi$ limits our model slightly, since in general the share space (domain of $C$) may contain more information than the current prices (cf. [1]). The bundle $r$ is determined by an agent's *demand function* $d(C, \pi)$ which specifies the bundle to buy given the price $\pi$ and the cost function $C$.

Our market dynamics are the following. The market maker posts the current price $\pi_t$, and at each time $t = 1 \dots T$, a trader is chosen with demand function $d$ drawn i.i.d. from some demand distribution $\mathcal{D}$. Intuitively, these demands are parameterized by latent variables such as the agent's belief $p \in \Delta_\Omega$ and total wealth $W$. The price is then updated to

$$\pi_{t+1} = \varphi(\varphi^{-1}(\pi_t) + d(C, \pi_t)). \tag{1}$$

After update $T$, the outcome is revealed and payout $\phi(\omega)_i$ is given for each contract $i \in \{1, \dots, n\}$.

## 3   Stationarity and equilibrium

We first would like to relate our stochastic model (1) to the standard notion of market equilibrium from the Economics literature, which we call the Walrasian equilibrium to avoid confusion. Here prices are fixed, and the equilibrium price is one that clears the market, meaning that the sum of the demands $r$ is $0 \in \mathbb{R}^n$. In fact, we will show that the stationary point of our process approaches the Walrasian equilibrium point as the liquidity of the market approaches infinity.

First, we must add a liquidity parameter to our market. Following the LMSR (the cost function $C(s) = b \ln \sum_i e^{s_i/b}$), we define

$$C_b(s) := b\,C(s/b). \tag{2}$$

This transformation of a convex function is called a *perspective function* and is known to preserve convexity [4]. Observe that $\varphi_b(s) := \nabla C_b(s) = \nabla C(s/b) = \varphi(s/b)$, meaning that the price under $C_b$ at $s$ is the same as the price under $C$ at $s/b$. As with the LMSR, we call $b$ the *liquidity parameter*; this terminology is justified by noting that one definition of liquidity, $1/\lambda_{\max}\nabla^2 C_b(s) = b/\lambda_{\max}\nabla^2 C(s/b)$ (cf. [1]). In the following, we will consider the limit as $b \to \infty$.

Second, in order to connect to the Walrasian equilibrium, we need a notion of a *fixed-price* demand function: if a trader has demand $d(C,\cdot)$ given $C$, what would the same trader's demand be under a market where prices are fixed and do not "change" during a trade? For the sake of generality, we restrict our allowable demand functions to the ones for which the limit

$$d(F,\pi) := \lim_{b\to\infty} d(C_b,\pi) \tag{3}$$

exists; this demand $d(F,\cdot)$ will be the corresponding *fixed-price demand* for $d$. We now define the Walrasion equilibrium point $\pi^*$, which is simply the price at which the market clears when traders have demands distributed by $\mathcal{D}$. Formally, this is the following condition:[1]

$$\int_{\mathcal{D}} d(F, \pi^*)\, d\mathcal{D}(d) = 0 \tag{4}$$

Note that $0 \in \mathbb{R}^n$; the demand for each contract should be balanced.

The stationary point of our stochastic process, on the other hand, is the price $\pi_b^s$ for which the expected price fluctuation is 0. Formally, we have

$$\mathop{\mathbb{E}}_{d\sim\mathcal{D}}[\Delta(\pi_b^s, d(C_b, \pi_b^s))] = 0, \tag{5}$$

where $\Delta(\pi, d) := \varphi(\varphi^{-1}(\pi) + d) - \pi$ is the price fluctuation. We now consider the limit of our stochastic process as the market liquidity approaches $\infty$.

**Theorem 1.** *Let $C$ be a strictly convex and $\alpha$-smooth[2] cost function, and assume that $\frac{\partial}{\partial b} d(C_b, \pi) = o(1/b)$ uniformly in $\pi$ and all $d \in \mathcal{D}$. If furthermore the limit (3) is uniform in $\pi$ and $d$, then $\lim_{b\to\infty} \pi_b^s = \pi^*$.*

*Proof.* Note that by the stationarity condition (5) we may define $\pi^*$ and $\pi_b^s$ to be the roots of the following "excess demand" functions, respectively:

$$Z(\pi) := \int_{\mathcal{D}} d(F,\pi)\, d\mathcal{D}(d), \qquad Z_b^s(\pi) := b \mathop{\mathbb{E}}_{d\sim\mathcal{D}}[\Delta(\pi, d(C_b,\pi))],$$

where we scale the latter by $b$ so that $Z_b^s$ does not limit to the zero function.

Let $s = \varphi^{-1}(\pi)$ be the current share vector. Then we have

$$\lim_{b\to\infty} b\Delta(\pi, d(C_b, \pi)) = \lim_{b\to\infty} b\left(\varphi\left(\varphi^{-1}(\pi) + d(C_b,\pi)/b\right) - \pi\right)$$

$$= \lim_{a\to 0} \frac{\varphi\left(s + a\,d(C_{1/a}, \pi)\right) - \pi}{a}$$

$$= \lim_{a\to 0} \nabla\varphi\left(s + a\,d(C_{1/a},\pi)\right)\left(d(C_{1/a},\pi) + a\tfrac{\partial}{\partial a}d(C_{1/a},\pi)\right)$$

$$= \lim_{b\to\infty} \nabla\varphi\left(s + \tfrac{1}{b} d(C_b,\pi)\right)\left(d(C_b,\pi) + \tfrac{1}{b}\tfrac{\partial}{\partial b}d(C_b,\pi)(-b^2)\right)$$

$$= \lim_{b\to\infty} \nabla^2 C(s)\,d(C_b,\pi) \;=\; \nabla^2 C(s)\,d(F,\pi),$$

where we apply L'Hopital's rule for the third equality. Crucially, the above limit is uniform with respect to both $d \in \mathcal{D}$ and $\pi \in \Pi$; uniformity in $d$ is by assumption, and uniformity in $\pi$ follows from $\alpha$-smoothness of $C$, since $C$ is dominated by a quadratic. Since the limit is uniform with respect to $\mathcal{D}$, we now have

$$\lim_{b \to \infty} Z_b^s(\pi) = \lim_{b \to \infty} b \mathop{\mathbb{E}}_{d \sim \mathcal{D}} [\Delta(\pi, d(C_b, \pi))] = \mathop{\mathbb{E}}_{d \sim \mathcal{D}} \left[ \lim_{b \to \infty} b \Delta(\pi, d(C_b, \pi)) \right]$$
$$= \nabla^2 C(s) \mathop{\mathbb{E}}_{d \sim \mathcal{D}} [d(F, \pi)] = \nabla^2 C(s) \, Z(\pi).$$

As $\nabla^2 C(s)$ is positive definite by assumption on $C$, we can conclude that $\lim_{b \to \infty} Z_b^s$ and $Z$ share the same zeroes. Since $Z$ has compact domain and is assumed continuous with a unique zero $\pi^*$, for each $\epsilon \in (0, \epsilon_{max})$ there must be some $\delta > 0$ s.t. $|Z(\pi)| > \epsilon$ for all $\pi$ s.t. $\|\pi - \pi^*\| > \delta$ (otherwise there would be a sequence of $\pi_n \to \pi'$ s.t. $f(\pi') = 0$ but $\pi' \neq \pi^*$). By uniform convergence there must be a $B > 0$ s.t. for all $b > B$ we have $\|Z_b^s - Z\|_\infty < \epsilon/2$. In particular, for $\pi$ s.t. $\|\pi - \pi^*\| > \delta$, $|Z_b^s(\pi)| > \epsilon/2$. Thus, the corresponding zeros $\pi_b^s$ must be within $\delta$ of $\pi^*$. Hence $\lim_{b \to \infty} \pi_b^s = \pi^*$.[3] $\qquad \square$

## 3.1 Utility-based demands

Maximum Expected Utility (MEU) demand functions are a particular kind of demand function derived by assuming a trader has some belief $p \in \Delta^n$ over the outcomes in $\Omega$, some wealth $W \geq 0$, and a monotonically increasing utility function of money $u : \mathbb{R} \to \mathbb{R}$. If such a trader buys a bundle $r$ of contracts from a market maker with cost function $C$ and price $\pi$, her wealth after $\omega$ occurs is $\Upsilon_\omega(C, W, \pi, r) := W + \phi(\omega) \cdot r - [C(\varphi^{-1}(\pi) + r) - C(\varphi^{-1}(\pi))]$. We ensure traders do not go into debt by requiring that traders only make demands such that this final wealth is nonnegative: $\forall \omega \, \Upsilon_\omega(C, \pi, r) \geq 0$. The set of debt-free bundles for wealth $W$ and market $C$ at price $\pi$ is denoted $S(C, W, \pi) := \{r \in \mathbb{R}^n \ : \ \min_\omega \Upsilon_\omega(C, W, \pi, r) \geq 0\}$.

A *continuous MEU demand function* $d_{W,p}^u(C, \pi)$ is then just the demand that maximizes a trader's expected utility subject to the debt-free constraint. That is,

$$d_{W,p}^u(C, \pi) := \operatorname*{argmax}_{r \in S(C,W,\pi)} \mathop{\mathbb{E}}_{\omega \sim p} [u(\Upsilon_\omega(C, W, \pi, r))]. \tag{6}$$

We also define a *fixed-price MEU demand function* $d_{W,p}^u(F, \pi)$ similarly, where $\Upsilon_\omega(F, W, \pi, r) := W + \phi(\omega) \cdot r - \pi \cdot r$ and $S(F, W, \pi) := \{r \in \mathbb{R}^n \ : \ \min_\omega \Upsilon_\omega(F, W, \pi, r) \geq 0\}$ are the fixed price analogues to the continuously priced versions above. Using the notation $bS := \{b \, r \, | \, r \in S\}$, the following relationships between the continuous and fixed price versions of $\Upsilon$, $S_W$, and the expected utility are a consequence of the convexity of $C$. Their main purpose is to highlight the relationship between wealth and liquidity in MEU demands. In particular, they show that scaling up of liquidity is equivalent to a scaling down of wealth and that the continuously priced constraints and wealth functions monotonically approach the fixed priced versions.

**Lemma 1.** *For any strictly convex cost function $C$, wealth $W > 0$, price $\pi$, demand $r$, and liquidity parameter $b > 0$ the following properties hold:    1. $\Upsilon_\omega(C_b, W, \pi, r) = b \, \Upsilon_\omega(C, W/b, \pi, r/b)$;    2. $S(C_b, W, \pi) = b \, S(C, W/b, \pi)$;    3. $S(C, W, \pi)$ is convex for all $C$;    4. $S(C, W, \pi) \subseteq S(C_b, W, \pi) \subseteq S(F, W, \pi)$ for all $b \geq 1$.    5. For monotone utilities $u$, $\mathbb{E}_{\omega \sim p} [u(\Upsilon_\omega(F, W, \pi, r))] \geq \mathbb{E}_{\omega \sim p} [u(\Upsilon_\omega(C, W, \pi, r))]$.*

*Proof.* Property (1) follows from a simple computation:

$$\Upsilon_\omega(C_b, W, \pi, r) = W + \phi(\omega) \cdot r - b \, C(\varphi^{-1}(\pi) + r/b) + b \, C(\varphi^{-1}(\pi))$$
$$= b \left( W/b + \phi(\omega) \cdot (r/b) - C(\varphi^{-1}(\pi) + r/b) + C(\varphi^{-1}(\pi)) \right),$$

which equals $b \, \Upsilon_\omega(C, W/b, \pi, r/b)$ by definition. We now can see property (2) as well:

$$S(C_b, W, \pi) = \{r \ : \ \min_\omega b \, \Upsilon_\omega(C, W/b, \pi, r/b) \geq 0\} = \{b \, r \ : \ \min_\omega \Upsilon_\omega(C, W/b, \pi, r) \geq 0\}.$$

For (3), define $f_{C,s,\omega}(r) = C(s+r) - C(s) - \phi(\omega) \cdot r$, which is the ex-post cost of purchasing bundle $r$. As $C$ is convex, and $f_{C,s,\omega}$ is a shifted and translated version of $C$ plus a linear term, $f_{C,s,\omega}$ is convex also. The constraint $\Upsilon_\omega(C, W, \pi, r) \geq 0$ then translates to $f_{C,s,\omega}(r) \leq W$, and thus the set of $r$ which satisfy the constraint is convex as a sublevel set of a convex function. Now $S(C, W, \pi)$ is convex as an intersection of convex sets, proving (3).

For (4) suppose $r$ satisfies $f_{C,s,\omega}(r) \leq W$. Note that $f_{C,s,\omega}(0) = 0$ always. Then by convexity we have for $f := f_{C,s,\omega}$ we have $f(r/b) = f\left(\frac{1}{b}r + \frac{b-1}{b}0\right) \leq \frac{1}{b}f(r) + \frac{b-1}{b}0 \leq W/b$, which implies $S(C, W, \pi) \subseteq S(C_b, W, \pi)$ when considering (3). To complete (4) note that $f_{C,s,\omega}$ dominates $f_{F,s} : r \mapsto (\varphi(s) - \phi(\omega)) \cdot r$ by convexity of $C$: $C(s+r) - C(s) \geq \nabla C(s) \cdot r$.

Finally, proof of (5) is obtained by noting that the convexity of $C$ means that $C(\varphi^{-1}(\pi) + r) - C(\varphi^{-1}(\pi)) \geq \nabla C(\varphi^{-1}(\pi)) \cdot r = \pi \cdot r$ and exploiting the monotonicty of $u$. $\square$

Lemma 1 shows us that MEU demands have a lot of structure, and in particular, properties (4) and (5) suggest that they may satisfy the conditions of Theorem 1; we leave this as an open question for future work. Another interesting aspect of Lemma 1 is the relationship between markets with cost function $C_b$ and wealths $W$ and markets with cost function $C$ and wealths $W/b$ – indeed, properties (1) and (2) suggest that the liquidity limit should in some sense be equivalent to a wealth limit, in that increasing liquidity by a factor $b$ should yield similar dynamics to decreasing the wealths by $b$. This would relate our model to that of [8], where the authors essentially show a wealth-limit version of Theorem 1 for a binary-outcome market where traders have linear utilities (a special case of (6)). We leave this precise connection for future work.

## 4   Market making as mirror descent

We now explore the surprising relationship between our stochastic price update and standard stochastic optimization techniques. In particular, we will relate our model to a stochastic *mirror descent* of the form

$$x_{t+1} = \underset{x \in \mathbb{R}}{\operatorname{argmin}}\{\eta\, x \cdot \nabla F(x_t; \xi) + D_R(x, x_t)\}, \tag{7}$$

where at each step $\xi \sim \Xi$ are i.i.d. and $R$ is some strictly convex function. We will refer to an algorithm of the form (7) a *stochastic mirror descent* of $f(x) := \mathbb{E}_{\xi \sim \Xi}[F(x; \xi)]$.

**Theorem 2.** *If for all $d \in \mathcal{D}$ we have some $F(\cdot; d) : \mathbb{R}^n \to \mathbb{R}^n$ such that $d(R^*, \pi) = -\nabla F(\pi; d)$, then the stochastic update of our model (1) is exactly a stochastic mirror descent of $f(\pi) = \mathbb{E}_{d \sim \mathcal{D}}[F(\pi; d)]$.*

*Proof.* By standard arguments, the mirror descent update (7) can be rewritten as

$$x_{t+1} = \nabla R^*(\nabla R(x_t) - \nabla F(x_t; \xi)),$$

where $R^*$ is the conjugate dual of $R$. Take $R = C^*$, and let $\xi = d \sim \mathcal{D}$. By assumption, we have $\nabla F(x; d) = -d(R^*, x) = -d(C, x)$ for all $d$. As $\nabla R^* = \nabla C = \varphi$, we have $\varphi^{-1} = (\nabla R^*)^{-1} = \nabla R$ by duality, and thus our update becomes $x_{t+1} = \varphi\left(\varphi^{-1}(x_t) + d(C, x_t)\right)$, which exactly matches the stochastic update of our model (1). $\square$

As an example, consider Kelly betters, which correspond to fixed-price demands $d(C, \pi) := d_{W,p}^{\log}(F, \pi)$ with utility $u(x) = \log x$ as defined in (3). A simple calculation shows that our update becomes

$$\pi_{t+1} = \varphi\left(\varphi^{-1}(\pi_t) + \frac{W}{\pi}\frac{p - \pi}{1 - \pi}\right), \tag{8}$$

where $W$ and $p$ are drawn (independently) from $\mathcal{P}$ and $\mathcal{W}$.

**Corollary 1.** *The stochastic update for fixed-price Kelly betters (8) is exactly a stochastic mirror descent of $f(\pi) = \overline{W} \cdot KL(\bar{p}, \pi)$, where $\bar{p}$ and $\overline{W}$ are the means of $\mathcal{P}$ and $\mathcal{W}$, respectively.*

*Proof.* We take $F(x; d_{W,p}^{\log}) = W \cdot (\text{KL}(p, x) + H(p))$. Then

$$\nabla F(x; d_{W,p}^{\log}) = W \left( \frac{-p}{x} + \frac{p-1}{1-x} \right) = -\frac{W}{x} \frac{p-x}{1-x} = -d_{W,p}^{\log}(F, x).$$

Hence, by Theorem 2 our update is a stochastic mirror descent of:

$$f(x) := \mathbb{E}[F(x; d_{W,p}^{\log})] = \mathbb{E}[Wp \log x + W(1-p) \log(1-x)] = \overline{W} \cdot (\text{KL}(\bar{p}, x) + H(\bar{p})),$$

which of course is equivalent to $\overline{W} \cdot \text{KL}(\bar{p}, x)$ as the entropy term does not depend on $x$. $\square$

Note that while this last result is quite compelling, we have mixed fixed-price demands with a continuous-price market model – see Section 3.1. One could interpret this combination as a model in which the market maker can only adjust the prices *after* a trade, according to a fixed convex cost function $C$. This of course differs from the standard model, which adjusts the price *continuously* during a trade.

## 4.1   Leveraging existing learning results

Theorem 2 not only identifies a fascinating connection between machine learning and our stochastic prediction market model, but it also allows us to use powerful existing techniques to make broad conclusions about the behavior of our model. Consider the following result:

**Proposition 1** ([6])**.** *If $\|\nabla F(\pi; p)\|^2 \leq G^2$ for all $p, \pi$, and $R$ is $\sigma$-strongly convex, then with probability $1 - \delta$,*

$$f(\bar{\pi}_T) \leq \min_\pi f(\pi) + \left( \frac{D^2}{\eta T} + \frac{G^2 \eta}{2\sigma} \right) \left( 1 + 4\sqrt{\log \frac{1}{\delta}} \right).$$

In our context, Proposition 1 says that the *average* of the prices will be a very good estimate of the minimizer of $f$, which as suggested by happens to be the underlying mean belief $\bar{p}$ of the traders! Moreover, as the Kelly demands are linear in both $p$ and $W$, it is easy to see from Theorem 1 that $\bar{p}$ is also the stationary point and the Walrasian equilibrium point (the latter was also shown by [11]). On the other hand, as we demonstrate next, it is not hard to come up with an example where the *instantaneous* price $\pi_t$ is quite far from the equilibrium at any given time period.

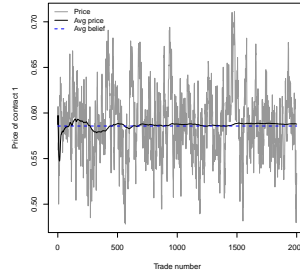

Figure 1: Price movement for Kelly betters with binomial($q = 0.6$, $n = 6$, $\alpha = 0.5$) beliefs in the LMSR market with liquidity $b = 10$.

Before moving to our empirical work, we make one final point. The above relationship between our stochastic market model and mirror descent sheds light on an important question: how might an automated market maker adjust the liquidity so that the market actually converges to the mean of the traders' beliefs? The learning parameter $\eta$ can be thought of as the inverse of the liquidity, and as such, Proposition 1 suggests that increasing the liquidity as $\sqrt{t}$ may cause the mean price to converge to the mean belief (assuming a fixed underlying belief distribution).

## 5   Empirical work

**Example: biased coin**   Consider a classic Bayesian setting where a coin has unknown bias $\Pr[\text{heads}] = q$, and traders have a prior $\beta(\alpha, \alpha)$ over $q$ (i.e., traders are $\alpha$-confident that the coin is fair). Now suppose each trader independently observes $n$ flips from the coin, and updates her belief; upon seeing $k$ heads, a trader would have posterior $\beta(\alpha + k, \alpha + n - k)$.

When presented with a prediction market with contracts for a single toss of the coin, where and contract 0 pays \$1 for tails and contract 1 pays \$1 for heads, a trader would purchase

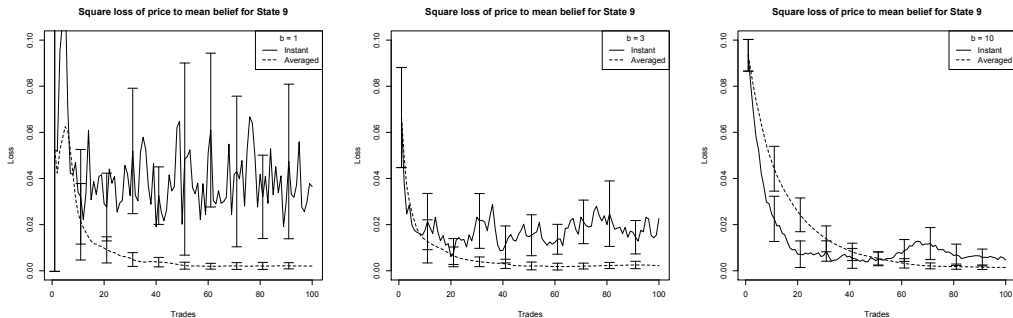

Figure 2: Mean square loss of average and instantaneous prices relative to the mean belief of 0.26 over 20 simulations for State 9 for $b = 1$ (left), $b = 3$ (middle), and $b = 10$ (right). Bars show standard deviation.

contracts as if according to the mean of their posterior. Hence, the belief distribution $\mathcal{P}$ of the market assigns weight $\mathcal{P}(p) = \binom{n}{k} q^k (1-q)^{n-k}$ to belief $p = (\alpha + k)/(2\alpha + n)$, yielding a biased mean belief of $(\alpha + nq)/(2\alpha + n)$.

We show a typical simulation of this market in Figure 1, where traders behave as Kelly betters in the fixed-price LMSR. Clearly, after almost every trade, the market price is quite far from the equilibrium/stationary point, and hence the classical supply and demand analysis of this market yields a poor description of the actual behavior, and in particular, of the predictive quality of the price at any given time. However, the mean price is consistently close to the mean belief of the traders, which in turn is quite close to the true parameter $q$.

**Election Survey Data**   We now compare the quality of the running average price versus the instantaneous price as a predictor of the mean belief of a market. We do so by simulating a market maker interacting with traders with unit wealth, log utility, and beliefs drawn from a fixed distribution. The belief distributions are derived from the Princeton election survey data[10]. For each of the 50 US states, participants in the survey were asked to estimate the probability that one of two possible candidates were going to win that state.[4] We use these 50 sets of estimates as 50 different empirical distributions from which to draw trader beliefs.

A simulation is configured by choosing one of the 50 empirical belief distributions $S$, a market liquidity parameter $b$ to define the LMSR cost function $C(s) = b \ln \sum_i e^{s_i/b}$, and an initial market position vector of $(0, 0)$ – that is, no contracts for either outcome. A configured simulation is run for $T$ trades. At each trade, a belief $p$ is drawn from $S$ uniformly and with replacement. This belief is used to determine the demand of the trader relative to the current market pricing. The trader purchase a bundle of contracts according to its demand and the market moves its position and price accordingly. The complete price path $\pi_t$ for $t = 1, \ldots, T$ of the market is recorded as well as a running average price $\bar{\pi}_t := \frac{1}{t} \sum_{i=1}^{t} \pi_t$ for $t = 1 \ldots, T$. For each of the 50 empirical belief distributions we configured 9 markets with $b \in \{1, 2, 3, 5, 10, 15, 20, 30, 50\}$ and ran 20 independent simulations of $T = 100$ trades. We present a portion of the results for the empirical distributions for states 9 and 11. States 9 and 11 have, respectively, sample sizes of 2,717 and 2,709; means 0.26 and 0.9; and variances 0.04 and 0.02. These are chosen as being representative of the rest of the simulation results: State 9 with mean off-center and a spread of beliefs (high uncertainty) and State 11 with highly concentrated beliefs around a single outcome (low uncertainty).

The results are summarised in Figures 2, 3, and 4. The first show the square loss of the average and instaneous prices relative to the mean belief for high uncertainty State 9 for $b = 1, 3, 10$. Clearly, the average price is a much more reliable estimator of the mean belief for low liquidity ($b = 1$) and is only outperformed by the instaneous price for higher liquidity ($b = 10$), but then only early in trading. Similar plots for State 11 are shown in Figure 3 where the advantage of using the average price is significantly diminished.

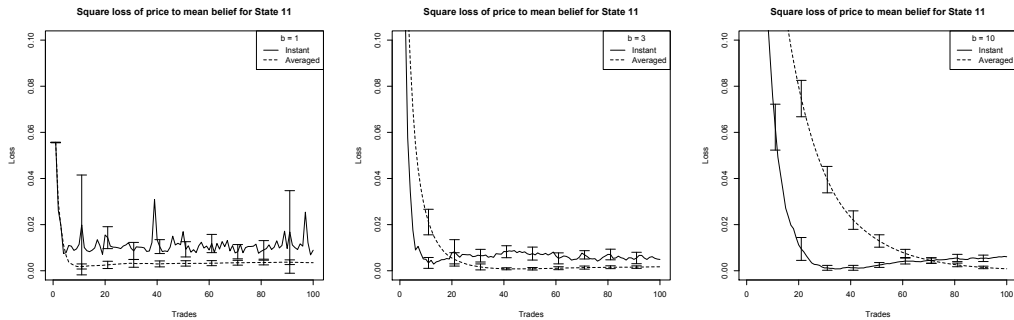

Figure 3: Mean square loss of average and instantaneous prices relative to the mean belief of 0.9 over 20 simulations for State 11 for $b = 1$ (left), $b = 3$ (middle), and $b = 10$ (right). Bars show standard deviation.

Figure 4 shows the improvement the average price has over the instantaneous price in square loss relative to the mean belief for all liquidity settings and highlights that average prices work better in low liquidity settings, consistent with the theory. Similar trends were observed for all the other States, depending on whether they had high uncertainty – in which case average price was a much better estimator – or low uncertainty – in which case instanteous price was better.

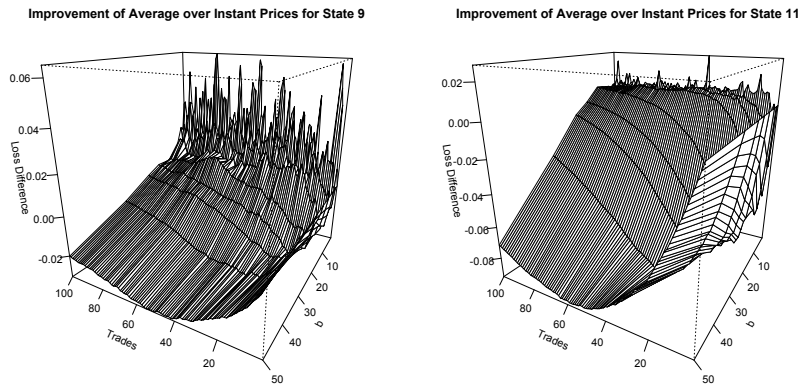

Figure 4: An overview of the results for States 9 (left) and 11 (right). For each trade and choice of $b$, the vertical value shows the improvement of the average price over the instantaneous price as measure by square loss relative to the mean.

## 6 Conclusion and future work

As noted in Section 3.1, there are several open questions with regard to maximum expected utility demands and Theorem 1, as well as the relationship between trader wealth and market liquidity. It would also be interesting to have a application of Theorem 2 to a continuous-price model, which yields a natural minimization as in Corollary 1. The equivalence to mirror decent stablished in Theorem 2 may also lead to a better understanding of the optimal manner in which a automated prediction market ought to increase liquidity so as to maximise efficiency.

**Acknowledgments**

This work was supported by the Australian Research Council (ARC). NICTA is funded by the Australian Government as represented by the Department of Broadband, Communications and the Digital Economy and the ARC through the ICT Centre of Excellence program. The first author was partially supported by NSF grant CC-0964033 and by a Google University Research Award.

## Footnotes

[1]Here and throughout we ignore technical issues of uniqueness. One may simply restrict to the class of demands for which uniqueness is satisfied.

[2]$C$ is $\alpha$-smooth if $\lambda_{\max}\nabla^2 C \le \alpha$

[3] We thank Avraham Ruderman for a helpful discussion regarding this proof.

[4]The original dataset contains conjunctions of wins as well as conditional statements but we only use the single variable results of the survey.

# References

[1] J. Abernethy, Y. Chen, and J.W. Vaughan. An optimization-based framework for automated market-making. *In Proceedings of the 11th ACM conference on Electronic Commerce (EC'11)*, 2011.

[2] A. Barbu and N. Lay. An introduction to artificial prediction markets for classification. *Arxiv preprint arXiv:1102.1465*, 2011.

[3] A. Beygelzimer, J. Langford, and D. Pennock. *Learning Performance of Prediction Markets with Kelly Bettors*. 2012.

[4] S. Boyd and L. Vandenberghe. *Convex optimization*. Cambridge University Press, 2004.

[5] Y. Chen and J.W. Vaughan. *A new understanding of prediction markets via no-regret learning*, pages 189–198. 2010.

[6] J. Duchi, S. Shalev-Shwartz, Y. Singer, and A. Tewari. Composite objective mirror descent. *COLT*, 2010.

[7] C.F. Manski. Interpreting the predictions of prediction markets. Technical report, National Bureau of Economic Research, 2004.

[8] A. Othman and T. Sandholm. When do markets with simple agents fail? In *Proceedings of the 9th International Conference on Autonomous Agents and Multiagent Systems: volume 1-Volume 1*, pages 865–872. International Foundation for Autonomous Agents and Multiagent Systems, 2010.

[9] A. Storkey. Machine learning markets. *AISTATS*, 2012.

[10] G. Wang, S.R. Kulkarni, H.V. Poor, and D.N. Osherson. Aggregating large sets of probabilistic forecasts by weighted coherent adjustment. *Decision Analysis*, 8(2):128, 2011.

[11] J. Wolfers and E. Zitzewitz. Interpreting prediction market prices as probabilities. Technical report, National Bureau of Economic Research, 2006.

